# Classification via Minimum Incremental Coding Length (MICL)

**John Wright**,* **Yi Ma**
Coordinated Science Laboratory
University of Illinois at Urbana-Champaign
{jnwright,yima}@uiuc.edu

**Yangyu Tao, Zhouchen Lin, Heung-Yeung Shum**
Visual Computing Group
Microsoft Research Asia
{v-yatao,zhoulin,hshum}@microsoft.com

## Abstract

We present a simple new criterion for classification, based on principles from lossy data compression. The criterion assigns a test sample to the class that uses the minimum number of additional bits to code the test sample, subject to an allowable distortion. We prove asymptotic optimality of this criterion for Gaussian data and analyze its relationships to classical classifiers. Theoretical results provide new insights into relationships among popular classifiers such as MAP and RDA, as well as unsupervised clustering methods based on lossy compression [13]. Minimizing the lossy coding length induces a regularization effect which stabilizes the (implicit) density estimate in a small-sample setting. Compression also provides a uniform means of handling classes of varying dimension. This simple classification criterion and its kernel and local versions perform competitively against existing classifiers on both synthetic examples and real imagery data such as handwritten digits and human faces, without requiring domain-specific information.

## 1 Introduction

One quintessential problem in statistical learning [9, 20] is to construct a classifier from labeled training data $(\boldsymbol{x}_i, y_i) \sim_{iid} p_{X,Y}(\boldsymbol{x}, y)$. Here, $\boldsymbol{x}_i \in \mathbb{R}^n$ is the observation, and $y_i \in \{1, \ldots, K\}$ its associated class label. The goal is to construct a classifier $g : \mathbb{R}^n \to \{1, \ldots, K\}$ which minimizes the expected risk (or probability of error): $g^* = \arg\min \mathbb{E}[I_{g(X) \neq Y}]$, where the expectation is taken with respect to $p_{X,Y}$. When the conditional class distributions $p_{X|Y}(\boldsymbol{x}|y)$ and the class priors $p_Y(y)$ are given, the *maximum a posterior* (MAP) assignment

$$\hat{y}(\boldsymbol{x}) = \arg\min_{y \in \{1,\ldots,K\}} -\ln p_{X|Y}(\boldsymbol{x}|y) - \ln p_Y(y) \tag{1}$$

gives the optimal classifier. This amounts to a *minimum coding length* principle: the optimal classifier minimizes the Shannon optimal (lossless) coding length of the test data $\boldsymbol{x}$ with respect to the distribution of the true class. The first term is the number of bits needed to code $\boldsymbol{x}$ w.r.t. the distribution of class $y$, and the second term is the number of bits needed to code the label $y$ for $\boldsymbol{x}$.

**Issues with Learning the Distributions from Training Samples.** In the typical classification setting, the distributions $p_{X|Y}(\boldsymbol{x}|y)$ and $p_Y(y)$ need to be learned from a set of labeled training

data. Conventional approaches to model estimation (implicitly) assume that the distributions are nondegenerate and the samples are sufficiently dense. However, these assumptions fail in many classification problems which are vital for applications in computer vision [10, 11]. For instance, the set of images of a human face taken from different angles and under different lighting conditions often lie in a low-dimensional subspace or submanifold of the ambient space [2]. As a result, the associated distributions are degenerate or nearly degenerate. Moreover, due to the high dimensionality of imagery data, the set of training images is typically sparse.

Inferring the generating probability distribution $p_{X,Y}$ from a sparse set of samples is an inherently ill-conditioned problem [20]. Furthermore, in the case of degenerate distributions, the classical likelihood function (1) does not have a well-defined maximum [20]. Thus, to infer the distribution from the training data or to use it to classify new observations, the distribution or its likelihood function needs to be properly "regularized." Typically, this is accomplished either explicitly via smoothness constraints, or implicitly via parametric assumptions on the distribution [3]. However, even if the distributions are assumed to be generic Gaussians, explicit regularization is still necessary to achieve good small-sample performance [6].

In many real problems in computer vision, the distributions associated with different classes of data have different model complexity. For instance, when detecting a face in an image, features associated with the face often have a low-dimensional structure which is "embedded" as a submanifold in a cloud of essentially random features from the background. Model selection criteria such as *minimum description length* (MDL) [12, 16] serve as important modifications to MAP for model estimation across classes of different complexity. It selects the model that minimizes the overall coding length of the given (training) data, hence the name "minimum description length" [1]. Notice, however, that MDL does not specify how the model complexity should be properly accounted for when classifying new test data among models that have different dimensions.

**Solution from Lossy Data Coding.** Given the difficulty of learning the (potentially degenerate) distributions $p_{X|Y}(\boldsymbol{x}|y)$ from a few samples in a high-dimensional space, it makes more sense to seek good "surrogates" for implementing the minimum coding length principle (1). Our idea is to measure how efficiently a new observation can be encoded by each class of the training data subject to an allowable distortion, and to assign the new observation to the class that requires the minimum number of additional bits. We dub this criterion "*minimum incremental coding length*" *(MICL)* for classification. It provides a counterpart of the MDL principle for model estimation and as a surrogate for the minimum coding length principle for classification.

The proposed MICL criterion naturally addresses the issues of regularization and model complexity. Regularization is introduced through the use of *lossy coding*, i.e. coding the test data $\boldsymbol{x}$ upto an allowable distortion[1] (placing our approach along the lines of lossy MDL [15]). This contrasts with Shannon's optimal *lossless* coding length, which requires precise knowledge of the true distributions. Lossy coding length also accounts for model complexity by directly measuring the difference in the volume (hence dimension) of the training data with and without the new observation.

**Relationships to Existing Classifiers.** While MICL and MDL both minimize a coding-theoretic objective, MICL differs strongly from traditional MDL approaches to classification such as those proven inconsistent in [8]. Those methods chose a *decision boundary* that minimizes the total number of bits needed to code the boundary and the samples it incorrectly classifies. In contrast, MICL uses coding length *directly* as a measure of how well the training data represent the new sample. The inconsistency result of [8] does not apply in this modified context. Within the lossy data coding framework, we establish that the MICL criterion leads to a family of classifiers that generalize the conventional MAP classifier (1). We prove that for Gaussian distributions, the MICL criterion asymptotically converges to a regularized version of MAP[2] (see Theorem 1) and give a precise estimate of the convergence rate (see Theorem 2). Thus, lossy coding induces a regularization effect similar to Regularized Discriminant Analysis (RDA) [6], with similar gains in finite sample performance with respect to MAP/QDA. The fully Bayesian approach to model estimation, in which posterior distributions over model parameters are estimated also provides finite sample gains over

ML/MAP [14]. However, that method is sensitive to the choice of prior when the number of samples is less than the dimension of the space, a situation that poses no difficulty to our proposed classifier.

When the distributions involved are not Gaussian, the MICL criterion can still be applied locally, similar to the popular k-Nearest Neighbor (k-NN) classifier. However, the local MICL classifier significantly improves the k-NN classifier as it accounts for both the number of samples and the distribution of the samples within the neighborhood. MICL can also be kernelized to handle nonlinear/non-Gaussian data, an extension similar to the generalization of Support Vector Machines (SVM) to nonlinear decision boundaries. The kernelized version of MICL provides a simple alternative to the SVM approach of constructing a linear decision boundary in the embedded (kernel) space, and better exploits the covariance structure of the embedded data.

## 2 Classification Criterion and Analysis

### 2.1 Minimum Incremental Coding Length.

A *lossy coding scheme* [5] maps vectors $\mathcal{X} = (\boldsymbol{x}_1, \ldots, \boldsymbol{x}_m) \in \mathbb{R}^{n \times m}$ to a sequence of binary bits, from which the original vectors can be recovered upto an allowable distortion $\mathbb{E}[\|\hat{\boldsymbol{x}} - \boldsymbol{x}\|^2] \le \varepsilon^2$. The length of the bit sequence is then a function $L_\varepsilon(\mathcal{X}) : \mathbb{R}^{n \times m} \to \mathbb{Z}_+$. If we encode each class of training data $\mathcal{X}_j \doteq \{\boldsymbol{x}_i : y_i = j\}$ separately using $L_\varepsilon(\mathcal{X}_j)$ bits, the entire training dataset can be represented by a two-part code using $\sum_{j=1}^K L_\varepsilon(\mathcal{X}_j) - |\mathcal{X}_j| \log_2 p_Y(j)$ bits. Here, the second term is the minimum number of bits needed to (losslessly) code the class labels $y_i$.

Now, suppose we are given a test observation $\boldsymbol{x} \in \mathbb{R}^n$, whose associated class label $y(\boldsymbol{x}) = j$ is unknown. If we code $\boldsymbol{x}$ jointly with the training data $\mathcal{X}_j$ of the $j$th class, the number of additional bits needed to code the pair $(\boldsymbol{x}, y)$ is $\delta L_\varepsilon(\boldsymbol{x}, j) = L_\varepsilon(\mathcal{X}_j \cup \{\boldsymbol{x}\}) - L_\varepsilon(\mathcal{X}_j) + L(j)$. Here, the first two terms measure the excess bits needed to code $(\boldsymbol{x}, \mathcal{X}_j)$ upto distortion $\varepsilon^2$, while the last term $L(j)$ is the cost of losslessly coding the label $y(\boldsymbol{x}) = j$. One may view these as "finite-sample lossy" surrogates for the Shannon coding lengths in the ideal classifier (1). This interpretation naturally leads to the following classifier:

**Criterion 1** (Minimum Incremental Coding Length). *Assign $\boldsymbol{x}$ to the class which minimizes the number of additional bits needed to code $(\boldsymbol{x}, \hat{y})$, subject to the distortion $\varepsilon$:*

$$\hat{y}(\boldsymbol{x}) \doteq \arg\min_{j \in \{1, \ldots, K\}} \delta L_\varepsilon(\boldsymbol{x}, j). \tag{2}$$

The above criterion (2) can be taken as a general principle for classification, in the sense that it can be applied using any lossy coding scheme. Nevertheless, effective classification demands that the chosen coding scheme be approximately optimal for the given data. From a finite sample perspective, $L_\varepsilon$ should approximate the Kolmogorov complexity of $\mathcal{X}$, while in an asymptotic, statistical setting it should approach the lower bound given by the rate-distortion of the generating distribution [5].

**Lossy Coding of Gaussian Data.** We will first consider a coding length function $L_\varepsilon$ introduced and rigorously justified in [13], which is (asymptotically) optimal for Gaussians. The (implicit) use of a coding scheme which is optimal for Gaussian sources is equivalent to assuming that the conditional class distributions $p_{X|Y}$ can be well-approximated by Gaussians. After rigorously analyzing this admittedly restrictive scenario, we will extend the MICL classifier (with this same $L_\varepsilon$ function) to arbitrary, multimodal distributions via an effective local Gaussian approximation.

For a multivariate Gaussian source $\mathcal{N}(\boldsymbol{\mu}, \Sigma)$, the average number of bits needed to code a vector subject to a distortion $\varepsilon^2$ is approximately $R_\varepsilon(\Sigma) \doteq \frac{1}{2} \log_2 \det\left(I + \frac{n}{\varepsilon^2}\Sigma\right)$ (bits/vector). Observations $\mathcal{X} = (\boldsymbol{x}_1, \ldots, \boldsymbol{x}_m)$ with sample mean $\hat{\boldsymbol{\mu}} = \frac{1}{m}\sum_i \boldsymbol{x}_i$ and covariance $\hat{\Sigma}(\mathcal{X}) = \frac{1}{m-1}\sum_i (\boldsymbol{x}_i - \hat{\boldsymbol{\mu}})(\boldsymbol{x}_i - \hat{\boldsymbol{\mu}})^T$ can be represented upto expected distortion $\varepsilon^2$ using $\approx m R_\varepsilon(\hat{\Sigma})$ bits. The optimal codebook is adaptive to the data, and can be encoded by representing the principal axes of the covariance using an additional $n R_\varepsilon(\hat{\Sigma})$ bits. Encoding the mean vector $\boldsymbol{\mu}$ requires an additional $\frac{n}{2} \log_2\left(1 + \frac{\hat{\boldsymbol{\mu}}^T\hat{\boldsymbol{\mu}}}{\varepsilon^2}\right)$ bits. The total number of bits required to code $\mathcal{X}$ is therefore

$$L_\varepsilon(\mathcal{X}) \doteq \frac{m+n}{2} \log_2 \det\left(I + \frac{n}{\varepsilon^2}\hat{\Sigma}(\mathcal{X})\right) + \frac{n}{2} \log_2\left(1 + \frac{\hat{\boldsymbol{\mu}}^T\hat{\boldsymbol{\mu}}}{\varepsilon^2}\right). \tag{3}$$

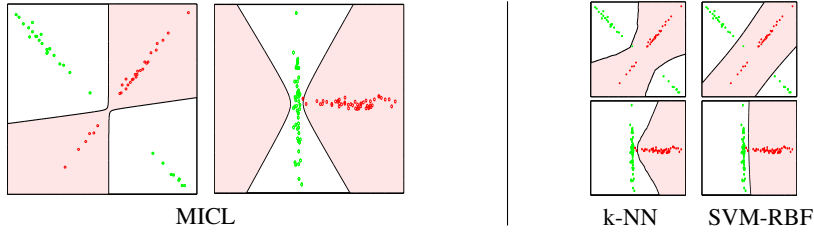

MICL                                          k-NN        SVM-RBF

Figure 1: MICL harnesses linear structure in the data to *interpolate* (left) and *extrapolate* (center) in sparsely sampled regions. Popular classifiers such as k-NN and SVM-RBF do not (right).

The first term gives the number of bits needed to represent the distribution of the $\boldsymbol{x}_i$ about their mean, and the second gives the cost of representing the mean. The above function well-approximates the optimal coding length for Gaussian data, and has also been shown to give a good upper bound on the number of bits needed to code finitely many samples lying on a linear subspace (e.g., a degenerate Gaussian distribution) [13].

**Coding the Class Label.** Since the label $Y$ is discrete, it can be coded losslessly. If the test class labels $Y$ are known to have the marginal distribution $P[Y = j] = \pi_j$, then the optimal coding lengths are (within one bit): $L(j) = -\log_2 \pi_j$. In practice, we may replace $\pi_j$ with the estimate $\hat{\pi}_j = |\mathcal{X}_j|/m$. Notice that as in the MAP classifier, the $\pi_j$ essentially form a prior on class labels. Combining this coding length the class label with the coding length function (3) for the observations, we summarize the MICL criterion (2) as Algorithm 1 below:

---

**Algorithm 1 (MICL Classifier).**

1: **Input:** $m$ training samples partitioned into $K$ classes $\mathcal{X}_1, \mathcal{X}_2, \ldots, \mathcal{X}_K$ and a test sample $\boldsymbol{x}$.
2: Compute prior distribution of class labels $\hat{\pi}_j = |\mathcal{X}_j|/m$.
3: Compute incremental coding length of $\boldsymbol{x}$ for each class:

$$\delta L_\varepsilon(\boldsymbol{x}, j) = L_\varepsilon(\mathcal{X}_j \cup \{\boldsymbol{x}\}) - L_\varepsilon(\mathcal{X}_j) - \log_2 \hat{\pi}_j,$$

where $\qquad L_\varepsilon(\mathcal{X}) \doteq \frac{m+n}{2}\log_2 \det\left(I + \frac{n}{\varepsilon^2}\hat{\Sigma}(\mathcal{X})\right) + \frac{n}{2}\log_2\left(1 + \frac{\hat{\boldsymbol{\mu}}^T\hat{\boldsymbol{\mu}}}{\varepsilon^2}\right).$

4: **Output:** $\hat{y}(\boldsymbol{x}) = \arg\min_{j=1,\ldots,K} \delta L_\varepsilon(\boldsymbol{x}, j).$

---

The $L_\varepsilon(\mathcal{X}_j \cup \{\boldsymbol{x}\})$ can be computed in $O(\min(m, n)^2)$ time (see [21]), allowing the MICL classifier to be directly applied to high-dimensional data. Figure 1 shows the performance of Algorithm 1 on two toy problems. In both cases, the MICL criterion harnesses the covariance structure of the data to achieve good classification in sparsely sampled regions. In the left example, the criterion *interpolates* the data structure to achieve correct classification, even near the origin where the samples are sparse. In the right example, the criterion *extrapolates* the horizontal line to the other side of the plane. Methods such as k-NN and SVM do not achieve the same effect. Notice, however, that these decision boundaries are similar to what MAP/QDA would give. This raises an important question: what is the precise relationship between MICL and MAP, and when is MICL superior?

## 2.2 Asymptotic Behavior and Relationship to MAP

In this section, we analyze the asymptotic behavior of Algorithm 1 as the number of training samples goes to infinity. The following result, whose proof is given in [21], indicates that MICL converges to a regularized version of ML/MAP, subject to a reward on the dimension of the classes:

**Theorem 1** (Asymptotic MICL [21]). *Let the training samples $\{(\boldsymbol{x}_i, y_i)\}_{i=1}^m \sim_{iid} p_{X,Y}(\boldsymbol{x}, y)$, with $\boldsymbol{\mu}_j \doteq \mathbb{E}[X|Y = j]$, $\Sigma_j \doteq Cov(X|Y = j)$. Then as $m \to \infty$, the MICL criterion coincides (asymptotically, with probability one) with the decision rule*

$$\hat{y}(\boldsymbol{x}) = \operatorname*{argmax}_{j=1,\ldots,K} \mathcal{L}_G\left(\boldsymbol{x} \,\middle|\, \boldsymbol{\mu}_j, \Sigma_j + \frac{\varepsilon^2}{n}I\right) + \ln \pi_j + \frac{1}{2}D_\varepsilon(\Sigma_j), \tag{4}$$

*where $\mathcal{L}_G(\cdot | \boldsymbol{\mu}, \Sigma)$ is the log-likelihood function for a $\mathcal{N}(\boldsymbol{\mu}, \Sigma)$ distribution, and $D_\varepsilon(\Sigma_j) \doteq \operatorname{tr}(\Sigma_j(\Sigma_j + \frac{\varepsilon^2}{n}I)^{-1})$ is the effective dimension of the $j$-th model, relative to the distortion $\varepsilon^2$.*

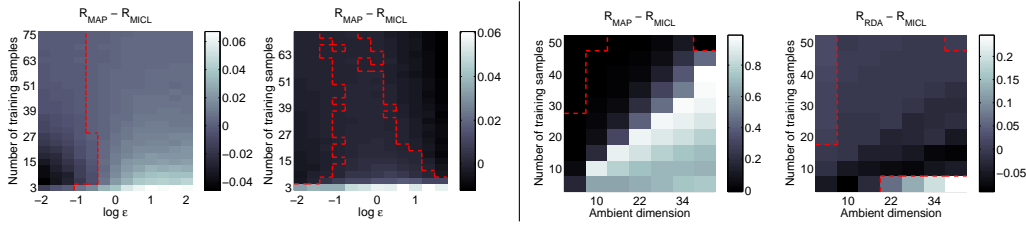

Figure 2: Left: Excess risk incurred by using MAP rather than MICL, as a function of $\varepsilon$ and $m$. (a) isotropic Gaussians. (b) anisotropic Gaussians. Right: Excess risk for nested classes, as a function of $n$ and $m$. (c) MICL vs. MAP. (d) MICL vs. RDA. In all examples, MICL is superior for $n \gg m$.

This result shows that asymptotically, MICL generates a family of MAP-like classifiers parametrized by the distortion $\varepsilon^2$. If all of the distributions are nondegenerate (i.e. their covariance matrices $\Sigma_j$ are nonsingular), then $\lim_{\varepsilon \to 0}(\Sigma_j + \frac{\varepsilon^2}{n}I) = \Sigma_j$ and $\lim_{\varepsilon \to 0} D_\varepsilon(\Sigma_j) = n$, a constant across the various classes. Thus, for nondegenerate data, the family of classifiers induced by MICL contains the conventional MAP classifier (1) at $\varepsilon = 0$. Given a finite number, $m$, of samples, any reasonable rule for choosing the distortion $\varepsilon^2$ should therefore ensure that $\varepsilon \to 0$ as $m \to \infty$. This guarantees that for non-degenerate distributions, MICL converges to the asymptotically optimal MAP criterion.

Simulations (e.g., Figure 1) suggest that the limiting behavior provides useful information even for finite training data. The following result, proven in [21], verifies that the MICL discriminant functions $\delta L_\varepsilon(\boldsymbol{x}, j)$ converge quickly to their limiting form $\delta L_\varepsilon^\infty(\boldsymbol{x}, j)$:

**Theorem 2** (MICL Convergence Rate [21]). *As the number of samples, $m \to \infty$, the MICL criterion* (2) *converges to its asymptotic form,* (4) *at a rate of $m^{-\frac{1}{2}}$. More specifically, with probability at least $1 - \alpha$, $\left| \delta L_\varepsilon(\boldsymbol{z}, j) - \delta L_\varepsilon^\infty(\boldsymbol{z}, j) \right| \leq c(\alpha) \cdot m^{-\frac{1}{2}}$ for some constant $c(\alpha) > 0$.*

### 2.3 Improvements over MAP

In the above, we have established the fact that asymptotically, the MICL criterion (4) is just as good as the MAP criterion. Nevertheless, the MICL criterion makes several important modifications to MAP, which significantly improve its performance on sparsely sampled or degenerate data.

**Regularization and Finite-Sample Behavior.** Notice that the first two terms of the asymptotic MICL criterion (4) have the form of a MAP criterion, based on an $\mathcal{N}(\boldsymbol{\mu}, \Sigma + \frac{\varepsilon^2}{n}I)$ distribution. This is somewhat equivalent to softening the distribution by $\frac{\varepsilon^2}{n}$ along each dimension, and has two important effects. First, it renders the associated MAP decision rule well-defined, even if the true data distribution is (almost) degenerate. Even for non-degenerate distributions, there is empirical evidence that for appropriately chosen $\varepsilon$, $\hat{\Sigma} + \frac{\varepsilon^2}{n}I$ gives more stable finite-sample classification [6].

Figure 2 demonstrates this effect on two simple examples. The generating distributions are parameterized as (a) $\boldsymbol{\mu}_1 = [-\frac{1}{2}, 0]$, $\boldsymbol{\mu}_2 = [\frac{1}{2}, 0]$, $\Sigma_1 = \Sigma_2 = I$, and (b) $\boldsymbol{\mu}_1 = [-\frac{3}{4}, 0]$, $\boldsymbol{\mu}_2 = [\frac{3}{4}, 0]$, $\Sigma_1 = diag(1, 4)$, $\Sigma_2 = diag(4, 1)$. In each example, we vary the number of training samples, $m$, and the distortion $\varepsilon$. For each $(m, \varepsilon)$ combination, we draw $m$ training samples from two Gaussian distributions $\mathcal{N}(\boldsymbol{\mu}_i, \Sigma_i), i = 1, 2$, and estimate the Bayes risk of the resulting MICL and MAP classifiers. This procedure is repeated 500 times, to estimate the overall Bayes risk with respect to variations in the training data. Figure 2 visualizes the difference in risks, $R_{MAP} - R_{MICL}$. Positive values indicate that MICL is outperforming MAP. The red line approximates the zero level-set, where the two methods perform equally well. In the isotropic case (a), MICL outperforms MAP for all sufficiently large $\varepsilon$. with a larger performance gain when the number of samples is small. In the anisotropic case (b), for most $\varepsilon$, MICL dramatically outperforms MAP for small sample sizes. We will see in the next example that this effect becomes more pronounced as the dimension increases.

**Dimension Reward.** The effective dimension term $D_\varepsilon(\Sigma_j)$ in the large-$n$ MICL criterion (4) can be rewritten as $D_\varepsilon(\Sigma_j) = \sum_{i=1}^n \lambda_i / (\frac{\varepsilon^2}{n} + \lambda_i)$, where $\lambda_i$ is the $i$th eigenvalue of $\Sigma_j$. If the data lie near a $d$-dimensional subspace ($\lambda_1 \ldots \lambda_d \gg \varepsilon^2/n$ and $\lambda_{d+1} \ldots \lambda_n \ll \varepsilon^2/n$), $D_\varepsilon \approx d$. In general,

$D_\varepsilon$ can be viewed as "softened" estimate of the dimension[3], relative to the distortion $\varepsilon^2$. MICL therefore rewards distributions that have relatively higher dimension.[4] However, this effect is somewhat countered by the regularization induced by $\varepsilon$, which rewards lower dimensional distributions.

Figure 2(right) empirically compares MICL to the conventional MAP and the *regularized* MAP (or RDA [6]). We draw $m$ samples from three nested Gaussian distributions: one of full rank $n$, one of rank $n/2$, and one of rank 1. All samples are corrupted by 4% Gaussian noise. We estimate the Bayes risk for each $(m, n)$ combination as in the previous example. The regularization parameter in RDA and the distortion $\varepsilon$ for MICL are chosen independently for each trial by cross validation. Plotted are the (estimated) differences in risk, $R_{MAP} - R_{MICL}$ (Fig. 2 (c)) and $R_{RDA} - R_{MICL}$ (Fig. 2 (d)). The red lines again correspond to the zero level-set of the difference. Unsurprisingly, MICL outperforms MAP for most $(m, n)$, and that the effect is most pronounced when $n$ is large and $m$ is small. When $m$ is much smaller than $n$ (e.g. the bottom row of Figure 2 right), MICL demonstrates a significant performance gain with respect to RDA. As the number of samples increases, there is a region where RDA is slightly better. For most $(m, n)$, MICL and RDA are close in performance.

### 2.4 Extensions to Non-Gaussian Data

In practice, the data distribution(s) of interest may not be Gaussian. If the rate-distortion function is known, one could, in principle, carry out similar analysis as for the Gaussian case. Nevertheless, in this subsection, we discuss two practical modifications to the MICL criterion that are applicable to arbitrary distributions and preserve the desirable properties discussed in the previous subsections.

**Kernel MICL Criterion.** Since $\mathcal{X}\mathcal{X}^T$ and $\mathcal{X}^T\mathcal{X}$ have the same non-zero eigenvalues,

$$\log_2 \det\!\left(I + \alpha\,\mathcal{X}\mathcal{X}^T\right) \;=\; \log_2 \det\!\left(I + \alpha\,\mathcal{X}^T\mathcal{X}\right). \tag{5}$$

This identity shows that $L_\varepsilon(\mathcal{X})$ can also be computed from the inner products between the $\boldsymbol{x}_i$. If the data $\boldsymbol{x}$ (of each class) are not Gaussian but there exists a nonlinear map $\psi : \mathbb{R}^n \to \mathcal{H}$ such that the transformed data $\psi(\boldsymbol{x})$ are (approximately) Gaussian, we can replace the inner product $\boldsymbol{x}_1^T\boldsymbol{x}_2$ with a symmetric positive definite *kernel function* $k(\boldsymbol{x}_1, \boldsymbol{x}_2) \doteq \psi(\boldsymbol{x}_1)^T\psi(\boldsymbol{x}_2)$. Choosing a proper kernel function will improve classification performance for non-Gaussian distributions. In practice, popular choices include the polynomial kernel $k(\boldsymbol{x}_1, \boldsymbol{x}_2) = (\boldsymbol{x}_1^T\boldsymbol{x}_2 + 1)^d$, the radial basis function (RBF) kernel $k(\boldsymbol{x}_1, \boldsymbol{x}_2) = \exp(-\gamma\|\boldsymbol{x}_1 - \boldsymbol{x}_2\|^2)$ and their variants. Implementation details, including how to properly account for the mean and dimension of the embedded data, are given in [21].

A similar transformation is used to generate nonlinear decision boundaries with SVM. Notice, however, that whereas SVM constructs a linear decision boundary in the lifted space $\mathcal{H}$, kernel MICL exploits the covariance structure of the lifted data, generating decision boundaries that are (asymptotically) quadratic. In Section 3 we will see that even for real data whose statistical nature is unclear, kernel MICL outperforms SVM when applied with the same kernel function.

**Local MICL Criterion.** For real data whose distribution is unknown, it may be difficult to find an appropriate kernel function. In this case, MICL can still be applied locally, in a neighborhood of the test sample $\boldsymbol{x}$. Let $N^k(\boldsymbol{x})$ denote the $k$ nearest neighbors of $\boldsymbol{x}$ in the training set $\mathcal{X}$. Training data in this neighborhood that belong to each class are $N_j^k(\boldsymbol{x}) \doteq \mathcal{X}_j \cap N^k(\boldsymbol{x}), j = 1, \ldots, K$. In the MICL classifier (Algorithm 1), we replace the incremental coding length $\delta L_\varepsilon(\boldsymbol{x}, j)$ by its local version:

$$\delta L_\varepsilon(\boldsymbol{x}, j) = L_\varepsilon(N_j^k(\boldsymbol{x}) \cup \{\boldsymbol{x}\}) - L_\varepsilon(N_j^k(\boldsymbol{x})) + L(j), \tag{6}$$

with $L(j) = -\log_2(|N_j^k(\boldsymbol{x})|/|N^k(\boldsymbol{x})|)$. Theorem 1 implies that this gives a universal classifier:

**Corollary 3.** *Suppose the conditional density $p_j(\boldsymbol{x}) = p(\boldsymbol{x}|y = j)$ of each class is nondegenerate. Then if $k = o(m)$ and $k, m \to \infty$, the local MICL criterion converges to the MAP criterion* (1).

This follows, since as the radius of the neighborhood shrinks, the cost of coding the class label, $-\log_2(|N_j^k(\boldsymbol{x})|/|N^k(\boldsymbol{x})|) \to -\log_2 p_j(\boldsymbol{x})$, dominates the coding length, (6). In this asymptotic setting the local MICL criterion behaves like k-Nearest Neighbor (k-NN). However, the finite-sample behavior of the local MICL criterion can differ drastically from that of k-NN, especially

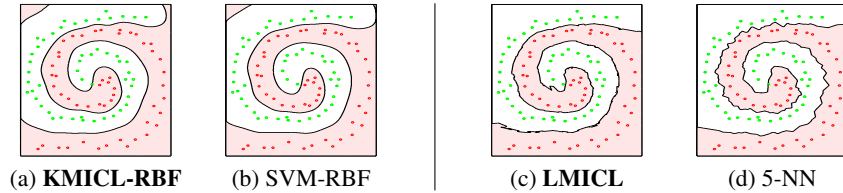

| | | |
|---|---|---|
| (a) **KMICL-RBF** | (b) SVM-RBF | (c) **LMICL** | (d) 5-NN |

Figure 3: Nonlinear extensions to MICL, compared to SVM and k-NN. Local MICL produces a smoother and more intuitive decision boundary than k-NN. Kernel MICL and SVM produce similar boundaries, that are smoother and better respect the data structure than those given by local methods.

| Method | Error | Method | Error | Method | Error | Method | Error |
|--------|-------|--------|-------|--------|-------|--------|-------|
| **LMICL** | **1.6%** | SVM-Poly [20] | 1.4% | **LMICL** | **4.9%** | **KMICL-Poly** | **4.7%** |
| k-NN | 3.1% | Best [18] | 0.4% | k-NN | 5.3% | SVM-Poly [4] | 5.3% |

Table 1: Results for handwritten digit recognition. Left: MNIST dataset. Right: USPS dataset, with identical preprocessing and kernel function. Here, kernel-MICL slightly outperforms SVM.

when the samples are sparse and the distributions involved are almost degenerate. In this case, from (4), local MICL effectively approximates the local shape of the distribution $p_j(\boldsymbol{x})$ by a (regularized) Gaussian, exploiting structure in the distribution of the nearest neighbors (see figure 3).

## 3  Experiments with Real Imagery Data

Using experiments on real data, we demonstrate that MICL and its nonlinear variants approach the best results from more sophisticated systems, without relying on domain-specific information.

**Handwritten Digit Recognition.**   We first test the MICL classifier on two standard datasets for handwritten digit recognition (Table 1 top). The MNIST handwritten digit dataset [10] consists of 60,000 training images and 10,000 test images. We achieved better results using the local version of MICL, due to non-Gaussian distribution of the data. With $k = 20$ and $\varepsilon = 150$, local MICL achieves a test error $1.59\%$, outperforming simple methods such as k-NN as well as many more complicated neural network approaches (e.g. LeNet-1 [10]). MICL's error rate approaches the best result for a generic learning machine ($1.4\%$ error for SVM with a degree-4 polynomial kernel). Problem-specific approaches have resulted in lower error rates, however, with the best reported result achieved using a specially engineered neural network [18].

We also test on the challenging USPS digits database (Table 1 bottom). Here, even humans have considerable difficulties ($\approx 2.5\%$ error). With $k = 35$ and $\varepsilon = 0.03$, local MICL achieves an error rate of $4.88\%$, again outperforming k-NN. We further compare the performance of kernel MICL to SVM (using [4]) on this dataset with the same homogeneous, degree 3 polynomial kernel, and identical preprocessing (normalization and centering), allowing us to compare pure classification performace. Here, SVM achieves a $5.3\%$ error, while kernel-MICL achieves an error rate of $4.7\%$ with distortion $\varepsilon = 0.0067$ (chosen automatically by cross-validation). Using domain-specific information, one can achieve better results. For instance [17] achieves $2.7\%$ error using tangent distance to a large number of prototypes. Other preprocessing steps, synthetic training images, or more advanced skew-correction and normalization techniques have been applied to lower the error rate for SVM (e.g. $4.1\%$ in [20]). While we have avoided extensive preprocessing here, so as to isolate the effect of the classifier, such preprocessing can be readily incorporated into our framework.

**Face Recognition.**   We further verify MICL's effectiveness on sparsely sampled high-dimensional data using the Yale Face Database B [7], which tests illumination sensitivity of face recognition algorithms. Following [7, 11], we use subsets 1 and 2 for training, and report the average test error across the four subsets. We apply Algorithm 1, *not* the local or kernel version, with $\varepsilon = 75$. MICL significantly outperforms classical subspace techniques on this problem (see Table 2), with error $0.9\%$ near the best reported results in [7, 11] that were obtained using a domain-specific model of

| Method | Error | Method | Error |
|--------|-------|--------|-------|
| **MICL** | **0.9%** | Eigenface [7] | 25.8% |
| Subspace [7] | 4.6% | Best [11] | 0% |



Subsets 1,2 (training)      Subsets 1-4 (testing)

Table 2: Face recognition under widely varying illumination. MICL outperforms classical face recognition methods such as Eigenfaces on Yale Face Database B [7].

illumination for face images. We suggest that the source of this improved performance is precisely the regularization induced by lossy coding. In this problem the number of training vectors per class, 19, is small compared to the dimension, $n = 32,256$ (for raw $168 \times 192$ images). Simulations (e.g. Figure 2) show that this is exactly the circumstance in which MICL is superior to MAP and even RDA. Interestingly, this suggests that directly exploiting degenerate or low-dimensional structures via MICL renders dimensionality reduction before classifying unnecessary or even undesirable.

## 4 Conclusion

We have proposed and studied a new information theoretic classification criterion, *Minimum Incremental Coding Length (MICL)*, establishing its optimality for Gaussian data. MICL generates a family of classifiers that inherit many of the good properties of MAP, RDA, and k-NN, while extending their working conditions to *sparsely sampled* or *degenerate* high-dimensional observations. MICL and its kernel and local versions approach best reported performance on high-dimensional visual recognition problems *without* domain-specific engineering. Due to its simplicity and flexibility, we believe MICL can be successfully applied to a wide range of real-world classification problems.

## Footnotes

*The authors gratefully acknowledge support from grants NSF Career IIS-0347456, NSF CRS-EHS-0509151, NSF CCF-TF-0514955, and ONR YIP N00014-05-1-0633.

[1]Information Bottleneck also uses lossy coding, but in an *unsupervised* manner, for clustering, feature selection and dimensionality reduction [19]. We apply lossy coding in the *supervised* (classification) setting.

[2]MAP subject to a Gaussian assumption is often referred to as Quadratic Discriminant Analysis (QDA) [9].

[3]This quantity has been dubbed the *effective number of parameters* in the context of ridge regression [9].

[4]This contrasts with the *dimension penalties* typical in model selection/estimation.

## References

[1] A. Barron, J. Rissanen, and B. Yu. The minimum description length principle in coding and modeling. *IEEE Transactions on Information Theory*, 44(6):2743–2760, 1998.

[2] R. Basri and D. Jacobs. Lambertian reflection and linear subspaces. *PAMI*, 25(2):218– 233, 2003.

[3] P. Bickel and B. Li. Regularization in statistics. *TEST*, 15(2):271–344, 2006.

[4] C. Chang and C. Lin. *LIBSVM: a library for support vector machines*, 2001.

[5] T. Cover and J. Thomas. *Elements of Information Theory*. Wiley Series in Telecommunications, 1991.

[6] J. Friedman. Regularized discriminant analysis. *JASA*, 84:165–175, 1989.

[7] A. Georghiades, P. Belhumeur, and D. Kriegman. From few to many: Illumination cone models for face recognition under variable lighting and pose. *PAMI*, 23(6):643–660, 2001.

[8] P. Grunwald and J. Langford. Suboptimal behaviour of Bayes and MDL in classification under misspecification. In *Proceedings of Conference on Learning Theory*, 2004.

[9] T. Hastie, R. Tibshirani, and J. Friedman. *The Elements of Statistical Learning*. Springer, 2001.

[10] Y. LeCun, L. Bottou, Y. Bengio, and P. Haffner. Gradient-based learning applied to document recognition. *Proceedings of the IEEE*, 86(11):2278–2324, 1998.

[11] K. Lee, J. Ho, and D. Kriegman. Acquiring linear subspaces for face recognition under variable lighting. *PAMI*, 27(5):684–698, 2005.

[12] J. Li. A source coding approach to classification by vector quantization and the principle of minimum description length. In *IEEE DCC*, pages 382–391, 2002.

[13] Y. Ma, H. Derksen, W. Hong, and J. Wright. Segmentation of multivariate mixed data via lossy data coding and compression. *PAMI*, 29(9):1546–1562, 2007.

[14] D. MacKay. Developments in probabilistic modelling with neural networks – ensemble learning. In *Proc. 3rd Annual Symposium on Neural Networks*, pages 191–198, 1995.

[15] M. Madiman, M. Harrison, and I. Kontoyiannis. Minimum description length vs. maximum likelihood in lossy data compression. In *IEEE International Symposium on Information Theory*, 2004.

[16] J. Rissanen. Modeling by shortest data description. *Automatica*, 14:465–471, 1978.

[17] P. Simard, Y. LeCun, and J. Denker. Efficient pattern recognition using a new transformation distance. In *Proceedings of NIPS*, volume 5, 1993.

[18] P. Simard, D. Steinkraus, and J. Platt. Best practice for convolutional neural networks applied to visual document analysis. In *ICDAR*, pages 958–962, 2003.

[19] N. Tishby, F. Pereira, and W. Bialek. The information bottleneck method. In *Allerton*, 1999.

[20] V. Vapnik. *The Nature of Statistical Learning Theory*. Springer, 2000.

[21] J. Wright, Y. Tao, Z. Lin, Y. Ma, and H. Shum. Classification via minimum incremental coding length (MICL). Technical report, UILU-ENG-07-2201, http://perception.csl.uiuc.edu/coding, 2007.

